# A Cortico-Cerebellar Model that Learns to Generate Distributed Motor Commands to Control a Kinematic Arm

**N.E. Berthier    S.P. Singh    A.G. Barto**
Department of Computer Science
University of Massachusetts
Amherst, MA 01002

**J.C. Houk**
Department of Physiology
Northwestern University Medical School
Chicago, IL 60611

## Abstract

A neurophysiologically-based model is presented that controls a simulated kinematic arm during goal-directed reaches. The network generates a quasi-feedforward motor command that is learned using training signals generated by corrective movements. For each target, the network selects and sets the output of a subset of pattern generators. During the movement, feedback from proprioceptors turns off the pattern generators. The task facing individual pattern generators is to recognize when the arm reaches the target and to turn off. A distributed representation of the motor command that resembles population vectors seen *in vivo* was produced naturally by these simulations.

## 1    INTRODUCTION

We have recently begun to explore the properties of sensorimotor networks with architectures inspired by the anatomy and physiology of the cerebellum and its interconnections with the red nucleus and the motor cortex (Houk 1989; Houk et al.,

1990). It is widely accepted that these brain regions are important in the control of limb movements (Kuypers, 1981; Ito, 1984), although relatively little attention has been devoted to probing how the different regions might function together in a cooperative manner. Starting from a foundation of known anatomical circuitry and the results of microelectrode recordings from neurons in these circuits, we proposed the concept of rubrocerebellar and corticocerebellar information processing modules that are arranged in parasagittal arrays and function as adjustable pattern generators (APGs) capable of the storage, recall and execution of motor programs.

The aim of the present paper is to extend the APG Model to a multiple degree-of-freedom task and to investigate how the motor representation developed by the model compares to the population vector representations seen by Georgopoulos and coworkers (e.g., Georopoulos, 1988). A complete description of the model and simulations reported here is contained in Berthier et al. (1991).

## 2    THE APG ARRAY MODEL

As shown in Figure 1 the model has three parts: a neural network that generates control signals, a muscle model that controls joint angle, and a planar, kinematic arm. The control network is an array of APGs that generate signals that are fed to the limb musculature. Because here we are interested in the basic issue of how a collection of APGs might cooperatively control multiple degree-of-freedom movements, we use a very simplified model of the limb that ignores dynamics. The muscles convert APG activity to changes in muscle length, which determine the changes in the joint angles. Activation of an APG causes movement of the arm in a direction in joint-angle space that is specific to that APG[1], and the magnitude of an APG's activity determines the velocity of that movement. The simultaneous activation of selected APGs determines the arm trajectory as the superposition of these movements. A learning rule, based on long-term depression (e.g., Ito, 1984), adjusts the subsets of APGs that are selected as well as characteristics of their activity in order to achieve desired movements.

Each APG consists of a positive feedback loop and a set of Purkinje cells (PCs). The positive feedback loop is a highly simplified model of a component of a complex cerebrocerebellar recurrent network. In the simplified model simulated here, each APG has its own feedback loop, and the loops associated with different APGs do not interact. When triggered by sufficiently strong activation, the neurons in these loops fire repetitively in a self-sustaining manner. An APG's motor command is generated through the action of its PCs which inhibit and modulate the buildup of activity in the feedback loop. The activity of loop cells is conveyed to spinal motor areas by rubrospinal fibers. PCs receive information that specifies and constrains the desired movements via parallel fibers.

We hypothesize that the response of PCs to particular parallel fiber inputs is adaptively adjusted through the influence of climbing fibers that respond to corrective movements (Houk & Barto, 1991). The APG array model assumes that climbing fibers and PCs are aligned in a way that climbing fibers provide specialized infor-

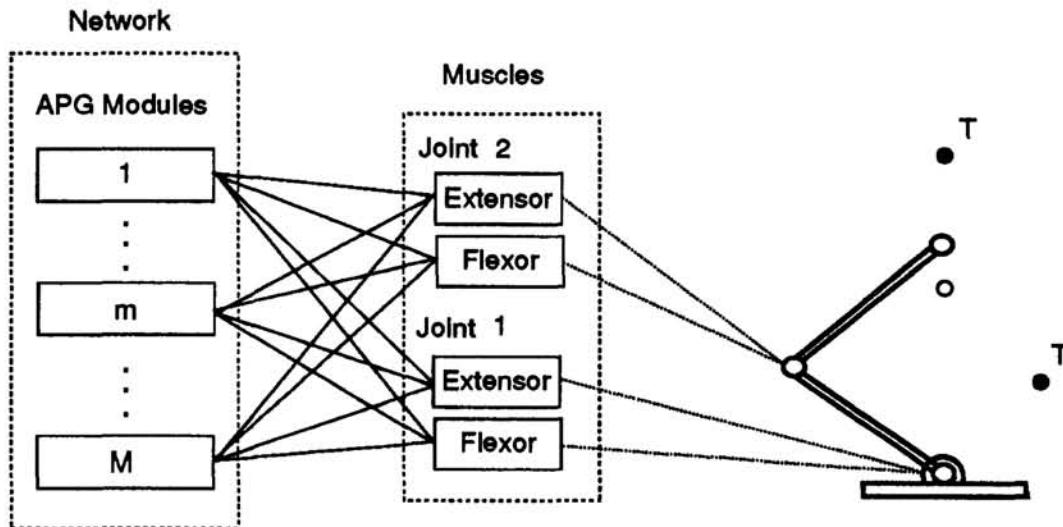

Figure 1: APG Control of Joint Angles. A collection of of APGs (adjustable pattern generators) is connected to a simulated two degree-of-freedom, kinematic, planar arm with antagonistic muscles at each joint. The task is to move the arm in the plane from a central starting location to one of eight symmetrically placed targets. Activation of an APG causes a movement of the arm that is specific to that APG, and the magnitude of an APG's activity determines the velocity of that movement. The simultaneous activation of selected APGs determines the arm trajectory as a superposition of these movements.

mation to PCs. Gellman et al. (1985) showed that proprioceptive climbing fibers are inhibited during planned movements, but the data of Gilbert and Thach (1977) suggest that they fire during corrective movements. In the present simulations, we assume that corrective movements are made when a movement fails to reach the target. These corrective movements stimulate proprioceptive climbing fibers which provides information to higher centers about the direction of the corrective movement. More detailed descriptions of APGs and relevant anatomy and physiology can be found in Houk (1989), Houk et al. (1990), and Berthier et al. (1991).

The generation of motor commands occurs in three phases. In the first phase, we assume that all positive feedback loops are off, and inputs provided by teleceptive and proprioceptive parallel fibers and basket cells determine the outputs of the PCs. We call this first phase selection. We assume that noise is present during the selection process so that individual PCs are turned off (i.e., selected) probabilistically. To begin the second phase, called the execution phase, loop activity is triggered by cortical activity. Once triggered, loop activity is self-sustaining because the loop cells have reciprocal positive connections. The triggering of loop activity causes the motor command to be "read out." The states of the PCs in the selection phase determine the speed and direction of the arm movement. As the movement is being performed, proprioceptive feedback and efference copy gradually depolarize the PCs. When a large proportion of the PCs are depolarized, PC inhibition reaches a critical value and terminates loop activity. In the third phase, the correction phase, corrective movements trigger climbing fiber activity that alters parallel fiber–PC connection weights.

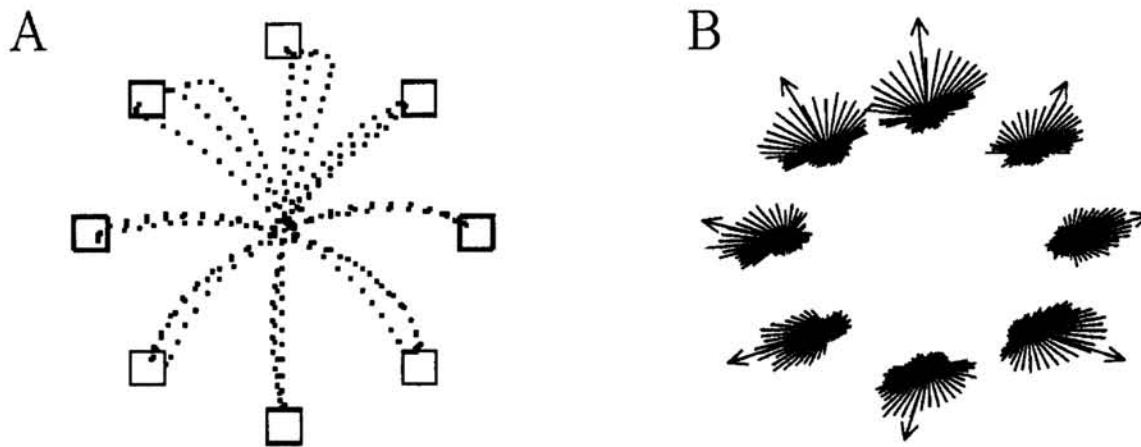

Figure 2: **A.** Movement Trajectories After Training. The starting point for each movement is the center of the workspace, and the target location is the center of the open square. The position of the arm at each time step is shown as a dot. Three movements are shown to each target. **B.** APG selection. APG selection for movements to a given target is illustrated by a vector plot at the position of the target. An individual APG is represented by a vector, the direction of which is equal to the direction of movement caused by that APG in Cartesian space. The vector length is proportional to output of the Purkinje cells during the selection phase. The arrow points in the direction of the vector sum.

## 3     SIMULATIONS

We trained the APG model to control a two degree-of-freedom, kinematic, planar arm. The task was similar to Georgopoulos (1988) and required APGs to move the arm from a central starting point to one of eight radially symmetric, equidistant targets. Each simulated trial started by placing the endpoint of the arm in the central starting location. The selection, execution, and correction phases of operation were then simulated. The task facing each of the selected APGs was to turn off at the proper time so that the movement stopped at the target.

Simulations showed that the model could learn to control movements to the eight targets. Training typically required about 700 trials per target until the arm endpoint was consistently moved to within 1 cm of the target. Figure 2 shows sample trajectories and population vectors of APG activity. Performance never resulted in precise movements due to the probabilistic nature of selection. Movement trajectories tended to follow straight lines in joint-angle space and were thus slightly curved lines in the workspace. About half of the APGs in the model were used to move to an individual target with population vectors similar to those seen by Georgopoulos (1988). The number of APGs used for each target was dependent on the sharpness of the climbing fiber receptive fields, with cardioid shaped receptive fields in joint-angle space giving population vectors that most resembled those experimentally observed.

# 4  ANALYSIS

In order to understand how the model worked we undertook a theoretical analysis of its simulated behavior. Analysis indicated that the expected trajectory of a movement was a straight line in joint-angle space from the starting position to the target. This is a special case of a mathematical result by Mussa-Ivaldi (1988). Because selection is probabilistic in the APG Array Model, trajectories in the workspace varied from the expected trajectory. In these cases, trajectories were piecewise linear because of the asynchronous termination of APG activity. Because of the Law of Large Numbers, the more PCs in each APG, the more closely the movement will resemble the expected movement.

The expected population of vectors of APG activity can be shown to be cosine-shaped in joint-angle space. That is, the length of the vector representing the activity of APG $m$ is proportional to the cosine of the angle between the direction of action of APG $m$ and the direction of the target in joint-angle space. The shape of the population vectors in Cartesian space is dependent on the Jacobian of the arm, which is a function of the arm posture.

The manner in which the outputs of PCs were set during selection leads to scaling of movement velocity with target distance. For any given movement direction, targets that are farther from the starting location lead to more rapid movements than closer targets.

Updating network weights based on the expected corrective movement will, in some cases, result in changing the weights in a way that they converge to the correct values. However, in other cases inappropriate changes are made. In the current simulations, we could largely avoid this problem by selecting parameter and initial weight values so that movements were initially small in amplitude. Random initialization of the weight values sometimes led to instances from which the learning rule could not recover.

# 5  DISCUSSION

In general, the present implementation of the model led to adequate control of the kinematic arm and mimicked the general output of nervous system seen in actual experiments. The network implemented a spatial to temporal transformation that transformed a target location into a time varying motor command. The model naturally generated population vectors that were similar to those seen *in vivo*. Further research is needed to improve the model's robustness and to extend it to more realistic control of a dynamical limb.

In the APG array model, APGs control arm movement in parallel so that the activity of all the modules taken together forms a distributed representation. The APG array executes a distributed motor program because it produces a spatiotemporal pattern of activity in the cerebrocerebellar recurrent network that is transmitted to the spinal cord to comprise a distributed motor command.

## 5.1   PARAMETRIZED MOTOR PROGRAMS

Certain features of the APG array model relate well to the ideas about parameterized motor programs discussed by Keele (1973), Schmidt (1988), and Adams (1971, 1977). The selection phase of the APG array model provides a feasible neuronal mechanism for preparing a parameterized motor program in advance of movement. The execution phase is also consistent with the open-loop ideas associated with motor programming concepts, except that, like Adams (1977), we explain the termination of the execution phase as being a consequence of proprioceptive feedback and efference copy.

In the APG array model, the counterpart of a generalized motor program is a set of parallel fiber weights for proprioceptive, efference copy, and target inputs. Given these weights, a particular constellation of parallel fiber inputs signifies that the desired endpoint of a movement is about to be reached, causing PCs to become depolarized. Once a set of parallel fiber weights corresponding to a desired endpoint is learned, the neuronal architecture and neurodynamics of the cerebellar network functions in a manner that parameterizes the motor program.

Movement velocity is parameterized in the selection phase of the model's operation. The velocity that is selected is automatically scaled so that velocity increases as the amplitude of the movement increases. While this type of scaling is often observed in motor performance studies, velocity can also be varied in an independent manner where velocity scaling can be applied simultaneously to all elements of a motor program to slow down or speed up the entire movement. Although we have not addressed this issue in the present report, simulation of velocity scaling under control of a neuromodulator can naturally be accomplished in the APG array model.

Movements terminate when the endpoint is recognized by PCs so that movement duration is dependent on the course of the movement instead of being determined by some internal clock because. Movement amplitude is parameterized by the weights of the target inputs, with smaller weights corresponding to larger amplitude movements.

## 5.2   CORRECTIVE MOVEMENTS

We assume that the training information conveyed to the APGs is the result of crude corrective movements stimulating proprioceptive receptors. This sensory information is conveyed to the cerebellum by climbing fibers. Learning in the APG array model therefore requires the existence of a low-level system capable of generating movements to spatial targets with at least a ballpark level of accuracy. Lesion (Yu et al., 1980) and developmental studies (von Hofsten, 1982) support the existence of a low-level system. Other evidence indicates that when limb movements are not proceeding accurately toward their intended targets, corrective components of the movements are generated by an unconscious, automatic control system (Goodale et al., 1986).

We assume that collaterals from the corticospinal and rubrospinal system that convey the motor commands to the spinal cord gate off sensory transmission through the proprioceptive climbing fiber pathway, thus preventing sensory responses to the initial limb movement. As the initial movement proceeds, the low-level system re-

ceives proprioceptive feedback from the limb and feedforward information about target location from the gaze control system. The latter information is updated as a consequence of corrective eye movements that typically occur after an initial gaze shift toward a visual target. Updated gaze information causes the spinal processor to generate a corrective component that is superimposed on the original motor command (Gielen & van Gisbergen, 1990; Flash & Henis, 1991). Since climbing fiber pathways would not be gated off by this low-level corrective process, climbing fibers should fire to indicate the direction of the corrective movement.

We assume that the network by which climbing fiber activity is generated is specifically wired to provide appropriate training information to the APGs (Houk & Barto, 1991). The training signal provided by a climbing fiber is specialized for the recipient APG in that it provides directional information in joint-angle space that is relative to the direction in which that APG moves the arm. The fact that training information is provided in terms of joint-angle space greatly simplifies the problem of providing errors in the correct system of reference. For example, if the network used visual error information, the error information would have to be transformed to joint errors.

The specialized training signals provided by the climbing fibers are determined by the structure of the ascending network conveying proprioceptive information. This ascending network has the same structure—but works in the opposite direction—as the network by which the APG array influences joint movement. This is reminiscent of the error backpropagation algorithm (e.g., Rumelhart et al., 1986, Parker, 1985) where the forward and backward passes through the network in the backpropagation algorithm are accomplished by the descending and ascending networks of the APG Array Model. This use of the ascending network to transform errors in the workspace to errors that are relative to a particular APG's direction of action is closely related to the use of error backpropagation for "learning with a distal teacher" as suggested by Jordan and Rumelhart (1991).

Houk and Barto (1991) suggested that the alignment of the ascending and descending networks might come about through trophic mechanisms stimulated by use-dependent alterations in synaptic efficacy. In the context of the present model, this hypothesis implies that the ascending network to the inferior olive, is established first, and that the descending network by which APGs influence motoneurons changes. We have not yet simulated this mechanism to see if it could actually generate the kind of alignment we assume in the present model.

### Acknowledgements

This research was supported by ONR N00014-88-K-0339, NIMH Center Grant P50 MH48185, and a grant from the McDonnell-Pew Foundation for Cognitive Neuroscience supported by the James S. McDonnell Foundation and the Pew Charitable Trusts.

## Footnotes

[1]To simplify these initial simulations we ignore changes in muscle moment arms with posture of the arm.

### References

Adams JA (1971) A closed-loop theory of motor learning. *J Motor Beh* 3: 111-149

Adams JA (1977) Feedback theory of how joint receptors regulate the timing and positioning of a limb. *Psychol Rev* 84: 504-523

Berthier NE Singh SP Barto AG Houk JC (1991) Distributed representation of limb motor programs in arrays of adjustable pattern generators. NPB Technical Report 3, Institute for Neuroscience, Northwestern University, Chicago IL

Flash T Henis E (1991) Arm trajectory modifications during reaching towards visual targets. *J Cognitive Neurosci* 3:220-230

Gellman R Gibson AR Houk JC (1985) Inferior olivary neurons in the awake cat: Detection of contact and passive body displacement. J Neurophys 54:40-60.

Georgopoulos A (1988) Neural integration of movement: role of motor cortex in reaching. FASEB Journal 2:2849-2857.

Gielen CCAM Gisbergen van JAM (1990) The visual guidance of saccades and fast aiming movements. *News in Physiol Sci* 5: 58-63

Gilbert PFC Thach WT (1977) Purkinje cell activity during motor learning. *Brain Res* 128:309-328.

Goodale MA Pelisson D Prablanc C (1986) Large adjustments in visually guided reaching do not depend on vision of the hand or perception of target displacement. *Nature* 320: 748-750

Hofsten von C (1982) Eye-hand coordination in the newborn. *Dev Psychol* 18: 450-461

Houk JC (1989) Cooperative control of limb movements by the motor cortex, brainstem and cerebellum. In: Cotterill RMJ (ed) Models of Brain Function. Cambridge Univ Press Cambridge UK, 309-325

Houk JC Barto AG (1991) Distributed sensorimotor learning. NPB Technical Report 1, Institute for Neuroscience, Northwestern University, Chicago IL

Houk JC Singh SP Fisher C Barto AG (1990) An adaptive sensorimotor network inspired by the anatomy and physiology of the cerebellum. In: Miller WT Sutton RS Werbos PJ (eds) Neural Networks for Control. MIT Press Cambridge, MA 301-348

Ito M (1984) The Cerebellum and Neural Control. Raven Press New York

Ito M (1989) Long-term depression. *Annual review of Neuroscience* 12: 85-102

Jordan MI Rumelhart DE (1991) Forward models: Supervised learning with a distal teacher. Occasional Paper #40 MIT Center for Cognitive Science

Keele SW (1973) Attention and Human Performance. Goodyear Pacific Palisades, California

Kuypers HGJM (1981) Anatomy of the descending pathways. In: Brooks VB (ed) Handbook of Physiology Section I Volume II Part 1. American Physiological Society Bethesda MD 597-666

Mussa-Ivaldi FA (1988) Do neurons in the motor cortex encode movement direction? An alternative hypothesis. *Neurosci Lett* 91:106-111

Parker DB (1985) Learning-Logic. Technical Report TR-47, Massachusetts Institute of Technology Cambridge MA

Rumelhart DE Hinton GE Williams RJ (1986) Learning internal representations by error propagation. In: Rumelhart DE McClelland JL (eds) Parallel Distributed Processing. Explorations in the Microstructure of Cognition, Vol. 1: Foundations. Bradford Books/MIT Press Cambridge MA

Schmidt RA (1988) Motor Control and Motor Learning. Human Kinetics Champaign, Illinois
